# Sparse Recovery with Brownian Sensing

**Alexandra Carpentier**
INRIA Lille
Alexandra.carpentier@inria.fr

**Odalric-Ambrym Maillard**
INRIA Lille
odalricambrym.maillard@gmail.com

**Rémi Munos**
INRIA Lille
remi.munos@inria.fr

## Abstract

We consider the problem of recovering the parameter $\alpha \in \mathbb{R}^K$ of a sparse function $f$ (i.e. the number of non-zero entries of $\alpha$ is small compared to the number $K$ of features) given noisy evaluations of $f$ at a set of well-chosen sampling points. We introduce an additional randomization process, called Brownian sensing, based on the computation of stochastic integrals, which produces a Gaussian sensing matrix, for which good recovery properties are proven, independently on the number of sampling points $N$, even when the features are arbitrarily non-orthogonal. Under the assumption that $f$ is Hölder continuous with exponent at least $1/2$, we provide an estimate $\widehat{\alpha}$ of the parameter such that $\|\alpha - \widehat{\alpha}\|_2 = O(\|\eta\|_2/\sqrt{N})$, where $\eta$ is the observation noise. The method uses a set of sampling points uniformly distributed along a one-dimensional curve selected according to the features. We report numerical experiments illustrating our method.

## 1 Introduction

We consider the problem of sensing an unknown function $f : \mathcal{X} \to \mathbb{R}$ (where $\mathcal{X} \subset \mathbb{R}^d$), where $f$ belongs to span of a large set of (known) features $\{\varphi_k\}_{1 \leq k \leq K}$ of $L_2(\mathcal{X})$:

$$f(x) = \sum_{k=1}^{K} \alpha_k \varphi_k(x),$$

where $\alpha \in \mathbb{R}^K$ is the unknown parameter, and is assumed to be $S$-sparse, i.e. $\|\alpha\|_0 \overset{\text{def}}{=} |\{i : \alpha_k \neq 0\}| \leq S$. Our goal is to recover $\alpha$ as accurately as possible.

In the setting considered here we are allowed to select the points $\{x_n\}_{1 \leq n \leq N} \in \mathcal{X}$ where the function $f$ is evaluated, which results in the noisy observations

$$y_n = f(x_n) + \eta_n, \tag{1}$$

where $\eta_n$ is an observation noise term. We assume that the noise is bounded, i.e., $\|\eta\|_2^2 \overset{\text{def}}{=} \sum_{n=1}^{N} \eta_n^2 \leq \sigma^2$. We write $\mathcal{D}_N = (\{x_n, y_n\}_{1 \leq n \leq N})$ the set of observations and we are interested in situations where $N \ll K$, i.e., the number of observations is much smaller than the number of features $\varphi_k$.

The question we wish to address is: how well can we recover $\alpha$ based on a set of $N$ noisy measurements? Note that whenever the noise is non-zero, the recovery cannot be perfect, so we wish to express the estimation error $\|\alpha - \widehat{\alpha}\|_2$ in terms of $N$, where $\widehat{\alpha}$ is our estimate.

**The proposed method.** We address the problem of sparse recovery by combining the two ideas:

- Sparse recovery theorems (see Section 2) essentially say that in order to recover a vector with a small number of measurements, one needs *incoherence*. The measurement basis, corresponding to the pointwise evaluations $f(x_n)$, should to be *incoherent* with the representation basis, corresponding to the one on which the vector $\alpha$ is sparse. Interpreting these basis in terms of linear operators, pointwise evaluation of $f$ is equivalent to measuring $f$ using Dirac masses $\delta_{x_n}(f) \stackrel{\text{def}}{=} f(x_n)$. Since in general the representation basis $\{\varphi_k\}_{1 \leq k \leq K}$ is not incoherent with the measurement basis induced by Dirac operators, we would like to consider another measurement basis, possibly randomized, in order that it becomes incoherent with any representation basis.

- Since we are interested in reconstructing $\alpha$, and since we assumed that $f$ is linear in $\alpha$, we can apply any set of $M$ linear operators $\{T_m\}_{1 \leq m \leq M}$ to $f = \sum_k \alpha_k \varphi_k$, and consider the problem transformed by the operators; the parameter $\alpha$ is thus also the solution to the transformed problem $T_m(f) = \sum_k \alpha_k T_m(\varphi_k)$.

Thus, instead of considering the $N \times K$ sensing matrix $\Phi = (\delta_{x_n}(\varphi_k))_{k,n}$, we consider a new $M \times K$ sensing matrix $A = (T_m(\varphi_k))_{k,m}$, where the operators $\{T_m\}_{1 \leq m \leq M}$ enforce incoherence between bases. Provided that we can estimate $T_m(f)$ with the data set $\mathcal{D}_N$, we will be able to recover $\alpha$. The *Brownian sensing* approach followed here uses stochastic integral operators $\{T_m\}_{1 \leq m \leq M}$, which makes the measurement basis incoherent with any representation basis, and generates a sensing matrix $A$ which is Gaussian (with i.i.d. rows).

The proposed algorithm (detailed in Section 3) recovers $\alpha$ by solving the system $A\alpha \approx \widehat{b}$ by $l_1$ minimization[1], where $\widehat{b} \in \mathbb{R}^M$ is an estimate, based on the noisy observations $y_n$, of the vector $b \in \mathbb{R}^M$ whose components are $b_m = T_m f$.

**Contribution:** Our contribution is a sparse recovery result for *arbitrary non-orthonormal* functional basis $\{\varphi_k\}_{k \leq K}$ of a Hölder continuous function $f$. Theorem 4 states that our estimate $\widehat{\alpha}$ satisfies $\|\alpha - \widehat{\alpha}\|_2 = O(\|\eta\|_2/\sqrt{N})$ with high probability *whatever* $N$, under the assumption that the noise $\eta$ is globally bounded, such as in [3, 12]. This result is obtained by combining two contributions:

- We show that when the sensing matrix $A$ is Gaussian, i.e. when each row of the matrix is drawn i.i.d. from a Gaussian distribution, orthonormality is not required for sparse recovery. This result, stated in Proposition 1 (and used in Step 1 of the proof of Theorem 4), is a consequence of Theorem 3.1 of [10].

- The sensing matrix $A$ is made Gaussian by choosing the operators $T_m$ to be stochastic integrals: $T_m f \stackrel{\text{def}}{=} \frac{1}{\sqrt{M}} \int_{\mathcal{C}} f dB^m$, where $B^m$ are Brownian motions, and $\mathcal{C}$ is a 1-dimensional curve of $\mathcal{X}$ appropriately chosen according to the functions $\{\varphi_k\}_{k \leq K}$ (see the discussion in Section 4). We call $A$ the *Brownian sensing* matrix.

We have the property that the recovery property using the Brownian sensing matrix $A$ only depends on the number of Brownian motions $M$ used in the stochastic integrals and not on the number of sampled points $N$. Note that $M$ can be chosen arbitrarily large as it is not linked with the limited amount of data, but $M$ affects the overall computational complexity of the method. The number of sample $N$ appears in the quality of estimation of $b$ only, and this is where the assumption that $f$ is Hölder continuous comes into the picture.

**Outline:** In Section 2, we survey the large body of existing results about sparse recovery and relate our contribution to this literature. In Section 3, we explain in detail the Brownian sensing recovery method sketched above and state our main result in Theorem 4.

In Section 4, we first discuss our result and compare it with existing work. Then we comment on the choice and influence of the sampling domain $\mathcal{C}$ on the recovery performance.

Finally in Section 5, we report numerical experiments illustrating the recovery properties of the Brownian sensing method, and the benefit of the latter compared to a straightforward application of compressed sensing when there is noise and very few sampling points.

## 2  Relation to existing results

A standard approach in order to recover $\alpha$ is to consider the $N \times K$ matrix $\Phi = (\varphi_k(x_n))_{k,n}$, and solve the system $\Phi \widehat{\alpha} \approx y$ where $y$ is the vector with components $y_n$. Since $N \ll K$ this is an ill-posed problem. Under the sparsity assumption, a successful idea is first to replace the initial problem with the well-defined problem of minimizing the $\ell_0$ norm of $\alpha$ under the constraint that $\Phi \widehat{\alpha} \approx y$, and then, since this problem is NP-hard, use convex relaxation of the $\ell_0$ norm by replacing it with the $\ell_1$ norm. We then need to ensure that the relaxation provides the same solution as the initial problem making use of the $\ell_0$ norm. The literature on this problem is huge (see [3, 7, 8, 15, 18, 4, 11] for examples of papers that initiated this field of research).

Generally, we can decompose the reconstruction problem into two distinct sub-problems. The first sub-problem (a) is to state conditions on the matrix $\Phi$ ensuring that the recovery is possible and derive results for the estimation error under such conditions:

The first important condition is the *Restricted Isometry Property* (RIP), introduced in [5], from which we can derive the following recovery result stated in [6]:

**Theorem 1 (Candés & al, 2006)** *Let $\delta_S$ be the restricted isometry constant of $\frac{\Phi}{\sqrt{N}}$, defined as $\delta_S = \sup\{|\frac{\|\frac{\Phi}{\sqrt{N}}a\|_2}{\|a\|_2} - 1|; \|a\|_0 \leq S\}$. Then if $\delta_{3S} + \delta_{4S} < 2$, for every $S$-sparse vector $\alpha \in \mathbb{R}^K$, the solution $\widehat{\alpha}$ to the $\ell_1$-minimization problem $\min\{\|a\|_1; a$ satisfies $\|\Phi a - y\|_2^2 \leq \sigma^2\}$ satisfies*

$$\|\widehat{\alpha} - \alpha\|_2^2 \leq \frac{C_S \sigma^2}{N},$$

*where $C_S$ depends only on $\delta_{4S}$.*

Apart from the historical RIP, many other conditions emerged from works reporting the practical difficulty to have the RIP satisfied, and thus weaker conditions ensuring reconstruction were derived. See [17] for a precise survey of such conditions. A weaker condition for recovery is the *compatibility condition* which leads to the following result from [16]:

**Theorem 2 (Van de Geer & Buhlmann, 2009)** *Assuming that the compatibility condition is satisfied, i.e. for a set $\mathcal{S}$ of indices of cardinality $S$ and a constant $L$,*

$$C(L, \mathcal{S}) = \min\left\{\frac{S\|\frac{\Phi}{\sqrt{N}}\alpha\|_2^2}{\|\alpha_{\mathcal{S}}\|_1^2}, \alpha \text{ satisfies } \|\alpha_{\mathcal{S}^c}\|_1 \leq L\|\alpha_{\mathcal{S}}\|_1\right\} > 0,$$

*then for every $S$-sparse vector $\alpha \in \mathbb{R}^K$, the solution $\widehat{\alpha}$ to the $\ell_1$-minimization problem $\min\{\|\alpha\|_1; \alpha$ satisfies $\|\alpha_{\mathcal{S}^c}\|_1 \leq L\|\alpha_{\mathcal{S}}\|_1\}$ satisfies for $C$ a numerical constant:*

$$\|\widehat{\alpha} - \alpha\|_2^2 \leq \frac{C}{C(L, \mathcal{S})^2} \frac{\sigma^2 \log(K)}{N}.$$

The second sub-problem (b) of the global reconstruction problem is to provide the user with a simple way to efficiently sample the space in order to build a matrix $\Phi$ such that the conditions for recovery are fulfilled, at least with high probability. This can be difficult in practice since it involves understanding the geometry of high dimensional objects. For instance, to the best of our knowledge, there is no result explaining how to sample the space so that the corresponding sensing matrix $\Phi$ satisfies the nice recovery properties needed by the previous theorems, for a *general* family of features $\{\varphi_k\}_{k \leq K}$.

However, it is proven in [12] that under some hypotheses on the functional basis, we are able to recover the strong RIP property for the matrix $\Phi$ with high probability. This result, combined with a recovery result, is stated as follows:

**Theorem 3 (Rauhut, 2010)** *Assume that $\{\varphi_k\}_{k \leq K}$ is an orthonormal basis of functions under a measure $\nu$, bounded by a constant $C_\varphi$, and that we build $\mathcal{D}_N$ by sampling $f$ at random according to $\nu$. Assume also that the noise is bounded $\|\eta\|_2 \leq \sigma$. If $\frac{N}{\log(N)} \geq c_0 C_\varphi^2 S \log(S)^2 \log(K)$ and $N \geq c_1 C_\varphi^2 S \log(p^{-1})$, then with probability at least $1 - p$, for every $S$-sparse vector $\alpha \in \mathbb{R}^K$, the solution $\widehat{\alpha}$ to the $\ell_1$-minimization problem $\min\{\|a\|_1; a$ satisfies $\|Aa - y\|_2^2 \leq \sigma^2\}$ satisfies*

$$\|\widehat{\alpha} - \alpha\|_2^2 \leq \frac{c_2 \sigma^2}{N},$$

*where $c_0$, $c_1$ and $c_2$ are some numerical constants.*

In order to prove this theorem, the author of [12] showed that by sampling the points i.i.d. from $\nu$, then with *with high probability* the resulting matrix $\Phi$ is RIP. The strong point of this Theorem is that we do not need to check conditions *on the matrix* $\Phi$ to guarantee that it is RIP, which is in practice infeasible. But the weakness of the result is that the initial basis has to be *orthonormal* and *bounded* under the given measure $\nu$ in order to get the RIP satisfied: the two conditions ensure incoherence with Dirac observation basis. The *specific* case of an unbounded basis i.e., Legendre Polynomial basis, has been considered in [13], but to the best of our knowledge, the problem of designing a *general* sampling strategy such that the resulting sensing matrix possesses nice recovery properties in the case of *non-orthonormal basis* remains unaddressed. Our contribution considers this case and is described in the following section.

## 3 The "Brownian sensing" approach

**A need for incoherence.** When the representation and observation basis are not incoherent, the sensing matrix $\Phi$ does not possess a nice recovery property. A natural idea is to change the observation basis by introducing a set of $M$ *linear operators* $\{T_m\}_{m \leq M}$ acting on the functions $\{\varphi_k\}_{k \leq K}$. We have $T_m(f) = \sum_{k=1}^{K} \alpha_k T_m(\varphi_k)$ for all $1 \leq m \leq M$ and our goal is to define the operators $\{T_m\}_{m \leq M}$ in order that the sensing matrix $(T_m(\varphi_k))_{m,k}$ enjoys a nice recovery property, whatever the representation basis $\{\varphi_k\}_{k \leq K}$.

**The Brownian sensing operators.** We now consider linear operators defined by stochastic integrals on a 1-dimensional curve $\mathcal{C}$ of $\mathcal{X}$. First, we need to select a curve $\mathcal{C} \subset \mathcal{X}$ of length $l$, such that the covariance matrix $V_{\mathcal{C}}$, defined by its elements $(V_{\mathcal{C}})_{i,j} = \int_{\mathcal{C}} \varphi_i \varphi_j$ (for $1 \leq i, j \leq K$), is invertible. We will discuss the existence of a such a curve later in Section 4. Then, we define the linear operators $\{T_m\}_{1 \leq m \leq M}$ as stochastic integrals over the curve $\mathcal{C}$: $T_m(g) \stackrel{\text{def}}{=} \frac{1}{\sqrt{M}} \int_{\mathcal{C}} g dB^m$, where $\{B^m\}_{m \leq M}$ are $M$ independent Brownian motions defined on $\mathcal{C}$.

Note that up to an appropriate speed-preserving parametrization $g : [0, l] \to \mathcal{X}$ of $\mathcal{C}$, we can work with the corresponding induced family $\{\psi_k\}_{k \leq K}$, where $\psi_k = \varphi_k \circ g$, instead of the family $\{\varphi_k\}_{k \leq K}$.

**The sensing method.** With the choice of the linear operators $\{T_m\}_{m \leq M}$ defined above, the parameter $\alpha \in \mathbb{R}^K$ now satisfies the following equation

$$A\alpha = b, \tag{2}$$

where $b \in \mathbb{R}^M$ is defined by its components $b_m \stackrel{\text{def}}{=} T_m(f) = \frac{1}{\sqrt{M}} \int_{\mathcal{C}} f(x) dB^m(x)$ and the so-called Brownian sensing matrix $A$ (of size $M \times K$) has elements $A_{m,k} \stackrel{\text{def}}{=} T_m(\varphi_k)$. Note that we do not require sampling $f$ in order to compute the elements of $A$. Thus, the samples only serve for estimating $b$ and for this purpose, we sample $f$ at points $\{x_n\}_{1 \leq n \leq N}$ regularly chosen along the curve $\mathcal{C}$.

In general, for a curve $\mathcal{C}$ parametrized with speed-preserving parametrization $g : [0, l] \to \mathcal{X}$ of $\mathcal{C}$, we have $x_n = g(\frac{n}{N}l)$ and the resulting estimate $\widehat{b} \in \mathbb{R}^M$ of $b$ is defined with components:

$$\widehat{b}_m = \frac{1}{\sqrt{M}} \sum_{n=0}^{N-1} y_n (B^m(x_{n+1}) - B^m(x_n)). \tag{3}$$

Note that in the special case when $\mathcal{X} = \mathcal{C} = [0, 1]$, we simply have $x_n = \frac{n}{N}$.

The final step of the proposed method is to apply standard recovery techniques (e.g., $l_1$ minimization or Lasso) to compute $\widehat{\alpha}$ for the system (2) where $b$ is perturbed by the so-called sensing noise $\varepsilon \stackrel{\text{def}}{=} b - \widehat{b}$ (estimation error of the stochastic integrals).

## 3.1 Properties of the transformed objects

We now give two properties of the Brownian sensing matrix $A$ and the sensing noise $\varepsilon = b - \widehat{b}$.

**Brownian sensing matrix.** By definition of the stochastic integral operators $\{T_m\}_{m \leq M}$, the sensing matrix $A = (T_m(\varphi_k))_{m,k}$ is a centered Gaussian matrix, with

$$\mathrm{Cov}(A_{m,k}, A_{m,k'}) = \frac{1}{M} \int_{\mathcal{C}} \varphi_k(x) \varphi_{k'}(x) dx \,.$$

Moreover by independence of the Brownian motions, each row $A_{m,\cdot}$ is i.i.d. from a centered Gaussian distribution $N(0, \frac{1}{M} V_{\mathcal{C}})$, where $V_{\mathcal{C}}$ is the $K \times K$ covariance matrix of the basis, defined by its elements $V_{k,k'} = \int_{\mathcal{C}} \varphi_k(x) \varphi_{k'}(x) dx$. Thanks to this nice structure, we can prove that $A$ possesses a property similar to RIP (in the sense of [10]) whenever $M$ is large enough:

**Proposition 1** *For $p > 0$ and any integer $t > 0$, when $M > \frac{C'}{4}(t \log(K/t) + \log 1/p)$, with $C'$ being a universal constant (defined in [14, 1]), then with probability at least $1 - p$, for all $t-$sparse vectors $x \in \mathbb{R}^K$,*

$$\frac{1}{2}\nu_{\min,\mathcal{C}}\|x\|_2 \leq \|Ax\|_2 \leq \frac{3}{2}\nu_{\max,\mathcal{C}}\|x\|_2,$$

*where $\nu_{\max,\mathcal{C}}$ and $\nu_{\min,\mathcal{C}}$ are respectively the largest and smallest eigenvalues of $V_{\mathcal{C}}^{1/2}$.*

**Sensing noise.** In order to state our main result, we need a bound on $\|\varepsilon\|_2^2$. We consider the simplest deterministic *sensing design* where we choose the sensing points to be uniformly distributed along the curve $\mathcal{C}$[2].

**Proposition 2** *Assume that $\|\eta\|_2^2 \leq \sigma^2$ and that $f$ is $(L, \beta)$-Hölder, i.e.*

$$\forall (x, y) \in \mathcal{X}^2, |f(x) - f(y)| \leq L|x - y|^{\beta} \,,$$

*then for any $p \in (0, 1]$, with probability at least $1 - p$, we have the following bound on the sensing noise $\varepsilon = b - \widehat{b}$:*

$$\|\varepsilon\|_2^2 \leq \frac{\tilde{\sigma}^2(N, M, p)}{N} \,,$$

*where*

$$\tilde{\sigma}^2(N, M, p) \stackrel{\text{def}}{=} 2\Big(\frac{L^2 l^{2\beta}}{N^{2\beta-1}} + \sigma^2\Big)\Big(1 + 2\frac{\log(1/p)}{M} + 4\sqrt{\frac{\log(1/p)}{M}}\Big) \,.$$

**Remark 1** *The bound on the sensing noise $\|\varepsilon\|_2^2$ contains two contributions: an approximation error term which comes from the approximation of a stochastic integral with $N$ points and that scales with $L^2 l^{2\beta}/N^{2\beta}$, and the observation noise term of order $\sigma^2/N$. The observation noise term (when $\sigma^2 > 0$) dominates the approximation error term whenever $\beta \geq 1/2$.*

## 3.2 Main result.

In this section, we state our main recovery result for the Brownian sensing method, described in Figure 1, using a uniform sampling method along a one-dimensional curve $\mathcal{C} \subset \mathcal{X} \subset \mathbb{R}^d$. The proof of the following theorem can be found in the supplementary material.

**Theorem 4 (Main result)** *Assume that $f$ is $(L, \beta)$-Hölder on $\mathcal{X}$ and that $V_{\mathcal{C}}$ is invertible. Let us write the condition number $\kappa_{\mathcal{C}} = \nu_{\max,\mathcal{C}}/\nu_{\min,\mathcal{C}}$, where $\nu_{\max,\mathcal{C}}$ and $\nu_{\min,\mathcal{C}}$ are respectively the largest and smallest eigenvalues of $V_{\mathcal{C}}^{1/2}$. Write $r = \left[(3\kappa_{\mathcal{C}} - 1)(\frac{1}{4\sqrt{2}-1})\right]^2$. For any $p \in (0, 1]$, let $M \geq 4c(4Sr \log(\frac{K}{4Sr}) + \log 1/p)$ (where $c$ is a universal constant defined in [14, 1]). Then, with probability at least $1 - 3p$, the solution $\widehat{\alpha}$ obtained by the Brownian sensing approach described in Figure 1, satisfies*

$$\|\widehat{\alpha} - \alpha\|_2^2 \leq C\Big(\frac{\kappa_{\mathcal{C}}^4}{\max_k \int_{\mathcal{C}} \varphi_k^2}\Big)\frac{\tilde{\sigma}^2(N, M, p)}{N} \,,$$

*where $C$ is a numerical constant and $\tilde{\sigma}(N, M, p)$ is defined in Proposition 2.*

Note that a similar result (not reported in this conference paper) can be proven in the case of i.i.d. sub-Gaussian noise, instead of a noise with bounded $\ell_2$ norm considered here.

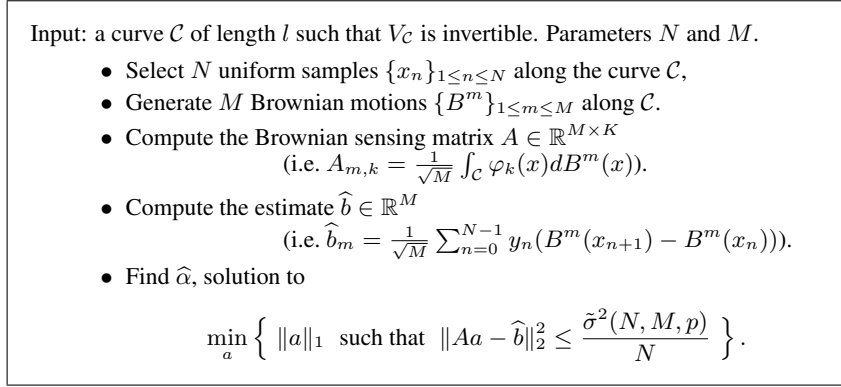

Input: a curve $\mathcal{C}$ of length $l$ such that $V_{\mathcal{C}}$ is invertible. Parameters $N$ and $M$.

- Select $N$ uniform samples $\{x_n\}_{1 \leq n \leq N}$ along the curve $\mathcal{C}$,
- Generate $M$ Brownian motions $\{B^m\}_{1 \leq m \leq M}$ along $\mathcal{C}$.
- Compute the Brownian sensing matrix $A \in \mathbb{R}^{M \times K}$
  (i.e. $A_{m,k} = \frac{1}{\sqrt{M}} \int_{\mathcal{C}} \varphi_k(x) dB^m(x)$).
- Compute the estimate $\widehat{b} \in \mathbb{R}^M$
  (i.e. $\widehat{b}_m = \frac{1}{\sqrt{M}} \sum_{n=0}^{N-1} y_n (B^m(x_{n+1}) - B^m(x_n))$).
- Find $\widehat{\alpha}$, solution to

$$\min_a \left\{ \|a\|_1 \quad \text{such that} \quad \|Aa - \widehat{b}\|_2^2 \leq \frac{\tilde{\sigma}^2(N,M,p)}{N} \right\}.$$

Figure 1: The Brownian sensing approach using a uniform sampling along the curve $\mathcal{C}$.

## 4 Discussion.

In this section we discuss the differences with previous results, especially with the work [12] recalled in Theorem 3. We then comment on the choice of the curve $C$ and illustrate examples of such curves for different bases.

### 4.1 Comparison with known results

**The order of the bound.** Concerning the scaling of the estimation error in terms of the number of sensing points $N$, Theorem 3 of [12] (reminded in Section 2) states that when $N$ is large enough (i.e., $N = \Omega(S \log(K))$), we can build an estimate $\widehat{\alpha}$ such that $\|\widehat{\alpha} - \alpha\|_2^2 = O(\frac{\sigma^2}{N})$. In comparison, our bound shows that $\|\widehat{\alpha} - \alpha\|_2^2 = O(\frac{L^2 l^{2\beta}}{N^{2\beta}} + \frac{\sigma^2}{N})$ *for any values* of $N$. Thus, provided that the function $f$ has a Hölder exponent $\beta \geq 1/2$, we obtain the same rate as in Theorem 3.

**A weak assumption about the basis.** Note that our recovery performance scales with the condition number $\kappa_{\mathcal{C}}$ of $V_{\mathcal{C}}$ as well as the length $l$ of the curve $\mathcal{C}$. However, concerning the hypothesis on the functions $\{\varphi_k\}_{k \leq K}$, we only assume that the covariance matrix $V_{\mathcal{C}}$ is invertible on the curve $\mathcal{C}$, which enables to handle *arbitrarily non-orthonormal bases*. This means that the orthogonality condition on the basis functions is not a crucial requirement to deduce sparse recovery properties. To the best of our knowledge, this is an improvement over previously known results (such as the work of [12]). Note however that if $\kappa_{\mathcal{C}}$ or $l$ are too high, then the bound becomes loose. Also the computational complexity of the Brownian sensing increases when $\kappa_{\mathcal{C}}$ is large, since it is necessary to take a large $M$, i.e. to simulate more Brownian motions in that case.

**A result that holds without any conditions on the number of sampling points.** Theorem 4 requires a constraint on the number of Brownian motions $M$ (i.e., that $M = \Omega(S \log K)$) and not on the number of sampling points $N$ (as in [12], see Theorem 3). This is interesting in practical situations when we do not know the value of $S$, as we do not have to assume a lower-bound on $N$ to deduce the estimation error result. This is due to the fact that the Brownian sensing matrix $A$ only depends on the computation of the $M$ stochastic integrals of the $K$ functions $\varphi_k$, and does not depend on the samples. The bound shows that we should take $M$ as large as possible. However, $M$ impacts the numerical cost of the method. This implies in practice a trade-off between a large $M$ for a good estimation of $\alpha$ and a low $M$ for low numerical cost.

### 4.2 The choice of the curve

**Why sampling along a $1$-dimensional curve $\mathcal{C}$ instead of sampling over the whole space $\mathcal{X}$?** In a bounded space $\mathcal{X}$ of dimension 1, both approaches are identical. But in dimension $d > 1$, following the Brownian sensing approach while sampling over the whole space would require generating $M$ Brownian sheets (extension of Brownian motions to $d > 1$ dimensions) over $\mathcal{X}$, and then building

the $M \times K$ matrix $A$ with elements $A_{m,k} = \int_{\mathcal{X}} \varphi_k(t_1, \ldots t_d) dB_1^m(t_1) \ldots dB_d^m(t_d)$. Assuming that the covariance matrix $V_{\mathcal{X}}$ is invertible, this Brownian sensing matrix is also Gaussian and enjoys the same recovery properties as in the one-dimensional case. However, in this case, estimating the stochastic integrals $b_m = \int_{\mathcal{X}} f dB^m$ using sensing points along a ($d$-dimensional) grid would provide an estimation error $\varepsilon = b - \widehat{b}$ that scales poorly with $d$ since we integrate over a $d$ dimensional space. This explains our choice of selecting a 1-dimensional curve $\mathcal{C}$ instead of the whole space $\mathcal{X}$ and sampling $N$ points along the curve. This choice provides indeed a better estimation of $b$ which is defined by a 1-dimensional stochastic integrals over $\mathcal{C}$. Note that the only requirement for the choice of the curve $\mathcal{C}$ is that the covariance matrix $V_{\mathcal{C}}$ defined along this curve should be invertible.

In addition, in some specific applications the sampling process can be very constrained by physical systems and sampling uniformly in all the domain is typically costly. For example in some medical experiments, e.g., scanner or I.R.M., it is only possible to sample along straight lines.

**What the parameters of the curve tell us on a basis.** In the result of Theorem 4, the length $l$ of the curve $\mathcal{C}$ as well as the condition number $\kappa_{\mathcal{C}} = \nu_{\max,\mathcal{C}}/\nu_{\min,\mathcal{C}}$ are essential characteristics of the efficiency of the method. It is important to note that those two variables are actually related. Indeed, it may not be possible to find a short curve $\mathcal{C}$ such that $\kappa_{\mathcal{C}}$ is small. For instance in the case where the basis functions have compact support, if the curve $\mathcal{C}$ does not pass through the support of all functions, $V_{\mathcal{C}}$ will not be invertible. Any function whose support does not intersect with the curve would indeed be an eigenvector of $V_{\mathcal{C}}$ with a 0 eigenvalue. This indicates that the method will not work well in the case of a very localized basis $\{\varphi_k\}_{k \leq K}$ (e.g. wavelets with compact support), since the curve would have to cover the whole domain and thus $l$ will be very large. On the other hand, the situation may be much nicer when the basis is not localized, as in the case of a Fourier basis. We show in the next subsection that in a $d$-dimensional Fourier basis, it is possible to find a curve $\mathcal{C}$ (actually a segment) such that the basis is orthonormal along the chosen line (i.e. $\kappa_{\mathcal{C}} = 1$).

## 4.3 Examples of curves

For illustration, we exhibit three cases for which one can easily derive a curve $\mathcal{C}$ such that $V_{\mathcal{C}}$ is invertible. The method described in the previous section will work with the following examples.

$\mathcal{X}$ **is a segment of** $\mathbb{R}$**:** In this case, we simply take $\mathcal{C} = \mathcal{X}$, and the sparse recovery is possible whenever the functions $\{\varphi_k\}_{k \leq K}$ are linearly independent in $\mathcal{L}_2$.

**Coordinate functions:** Consider the case when the basis are the coordinate functions $\varphi_k(t_1, \ldots t_d) = t_k$. Then we can define the parametrization of the curve $\mathcal{C}$ by $g(t) = \alpha(t)(t, t^2, \ldots, t^d)$, where $\alpha(t)$ is the solution to a differential equation such that $\|g'(t)\|_2 = 1$ (which implies that for any function $h$, $\int h \circ g = \int_{\mathcal{C}} h$). The corresponding functions $\psi_k(t) = \alpha(t) t^k$ are linearly independent, since the only functions $\alpha(t)$ such that the $\{\psi_k\}_{k \leq K}$ are not linearly independent are functions that are 0 almost everywhere, which would contradict the definition of $\alpha(t)$. Thus $V_{\mathcal{C}}$ is invertible.

**Fourier basis:** Let us now consider the Fourier basis in $\mathbb{R}^d$ with frequency $T$:

$$\varphi_{n_1, \ldots, n_d}(t_1, \ldots, t_d) = \prod_j \exp\big(-\frac{2i\pi n_j t_j}{T}\big),$$

where $n_j \in \{0, \ldots, T-1\}$ and $t_j \in [0, 1]$. Note that this basis is orthonormal under the uniform distribution on $[0, 1]^d$. In this case we define $g$ by $g(t) = \lambda(t\frac{1}{T^{d-1}}, t\frac{T}{T^{d-1}}, \ldots, t\frac{T^{d-1}}{T^{d-1}})$ with $\lambda = \sqrt{\frac{1-T^{-2}}{1-T^{-2d}}}$ (so that $\|g'(t)\|_2 = 1$), thus we deduce that:

$$\psi_{n_1, \ldots, n_d}(t) = \exp\big(-\frac{2i\pi t\lambda \sum_j n_j T^{j-1}}{T^d}\big).$$

Since $n_k \in \{0, \ldots, T-1\}$, the mapping that associates $\sum_j n_j T^{j-1}$ to $(n_1, \ldots, n_d)$ is a bijection from $\{0, \ldots, T-1\}^d$ to $\{0, \ldots, T^d-1\}$. Thus we can identify the family $(\psi_{n_1, \ldots, n_d})$ with the one dimensional Fourier basis with frequency $\frac{T^d}{\lambda}$, which means that the condition number $\rho = 1$ for this curve. Therefore, for a $d$-dimensional function $f$, sparse in the Fourier basis, it is sufficient to sample along the curve induced by $g$ to ensure that $V_{\mathcal{C}}$ is invertible.

# 5 Numerical Experiments

In this section, we illustrate the method of Brownian sensing in dimension one. We consider a non-orthonormal family $\{\varphi_k\}_{k \leq K}$ of $K = 100$ functions of $L_2([0, 2\pi])$ defined by $\varphi_k(t) = \frac{\cos(tk) + \cos(t(k+1))}{\sqrt{2\pi}}$. In the experiments, we use a function $f$ whose decomposition is 3-sparse and which is $(10, 1)$-Hölder, and we consider a bounded observation noise $\eta$, with different noise levels, where the noise level is defined by $\sigma^2 = \sum_{n=1}^{N} \eta_n^2$.

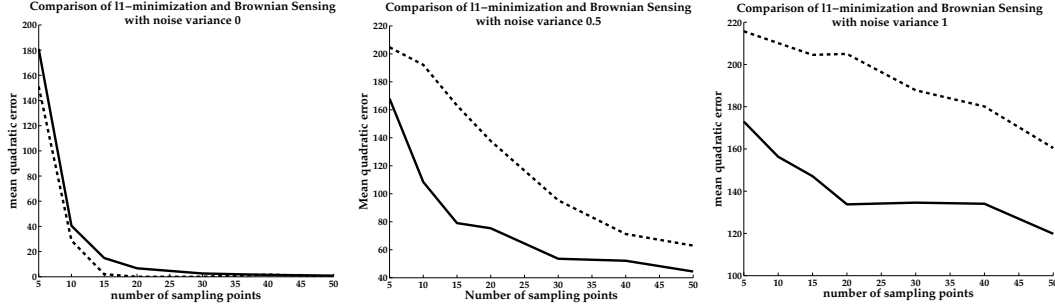

Figure 2: Mean squared estimation error using Brownian sensing (plain curve) and a direct $l_1$-minimization solving $\Phi\alpha \approx y$ (dashed line), for different noise level ($\sigma^2 = 0$, $\sigma^2 = 0.5$, $\sigma^2 = 1$), plotted as a function of the number of sample points $N$.

In Figure 2, the plain curve represents the recovery performance, i.e., mean squared error, of Brownian sensing i.e., minimizing $\|a\|_1$ under constraint that $\|Aa - \widehat{b}\|_2 \leq 1.95\sqrt{2(100/N + 2)}$ using $M = 100$ Brownian motions and a regular grid of $N$ points, as a function of $N$[3]. The dashed curve represents the mean squared error of a regular $l_1$ minimization of $\|a\|_1$ under the constraint that $\|\Phi a - y\|_2^2 \leq \sigma^2$ (as described e.g. in [12]), where the $N$ samples are drawn uniformly randomly over the domain. The three different graphics correspond to different values of the noise level $\sigma^2$ (from left to right 0, 0.5 and 1). Note that the results are averaged over 5000 trials.

Figure 2 illustrates that, as expected, Brownian sensing outperforms the method described in [12] for noisy measurements[4]. Note also that the method described in [12] recovers the sparse vector when there is no noise, and that Brownian sensing in this case has a smoother dependency w.r.t. $N$. Note that this improvement comes from the fact that we use the Hölder regularity of the function: Compressed sensing may outperform Brownian sensing for arbitrarily non regular functions.

## Conclusion

In this paper, we have introduced a so-called Brownian sensing approach, as a way to sample an unknown function which has a sparse representation on a given non-orthonormal basis. Our approach differs from previous attempts to apply compressed sensing in the fact that we build a "Brownian sensing" matrix $A$ based on a set of Brownian motions, which is independent of the function $f$. This enables us to guarantee nice recovery properties of $A$. The function evaluations are used to estimate the right hand side term $b$ (stochastic integrals). In dimension $d$ we proposed to sample the function along a well-chosen curve, i.e. such that the corresponding covariance matrix is invertible. We provided competitive reconstruction error rates of order $O(\|\eta\|_2/\sqrt{N})$ when the observation noise $\eta$ is bounded and $f$ is assumed to be Hölder continuous with exponent at least $1/2$. We believe that the Hölder assumption is not strictly required (the smoothness of $f$ is assumed to derive nice estimations of the stochastic integrals only), and future works will consider weakening this assumption, possibly by considering randomized sampling designs.

**Acknowledgements**

This research was partially supported by the French Ministry of Higher Education and Research, Nord- Pas-de-Calais Regional Council and FEDER through CPER 2007-2013, ANR projects EXPLO-RA (ANR-08-COSI-004) and Lampada (ANR-09-EMER-007), by the European Communitys Seventh Framework Programme (FP7/2007-2013) under grant agreement 231495 (project CompLACS), and by Pascal-2.

## Footnotes

[1]where the approximation sign $\approx$ refers to a minimization problem under a constraint coming from the observation noise.

[2]Note that other deterministic, random, or low-discrepancy sequence could be used here.

[3]We assume that we know a loose bound on the noise level, here $\sigma^2 \leq 2$, and we take $p = 0.01$.

[4]Note however that there is no theoretical guarantee that the method described in [12] works here since the functions are not orthonormal.

# References

[1] R. Baraniuk, M. Davenport, R. DeVore, and M. Wakin. A simple proof of the restricted isometry property for random matrices. *Constructive Approximation*, 28(3):253–263, 2008.

[2] G. Bennett. Probability inequalities for the sum of independent random variables. *Journal of the American Statistical Association*, 57(297):33–45, 1962.

[3] E. Candès and J. Romberg. Sparsity and incoherence in compressive sampling. *Inverse Problems*, 23:969–985, 2007.

[4] E. Candes and T. Tao. The Dantzig selector: statistical estimation when p is much larger than n. *Annals of Statistics*, 35(6):2313–2351, 2007.

[5] E.J. Candès, J. Romberg, and T. Tao. Robust uncertainty principles: Exact signal reconstruction from highly incomplete frequency information. *IEEE Transactions on information theory*, 52(2):489–509, 2006.

[6] E.J. Candès, J.K. Romberg, and T. Tao. Stable signal recovery from incomplete and inaccurate measurements. *Communications on Pure and Applied Mathematics*, 59(8):1207, 2006.

[7] D.L. Donoho. Compressed sensing. *IEEE Transactions on Information Theory*, 52(4):1289–1306, 2006.

[8] D.L. Donoho and P.B. Stark. Uncertainty principles and signal recovery. *SIAM Journal on Applied Mathematics*, 49(3):906–931, 1989.

[9] M. Fornasier and H. Rauhut. Compressive Sensing. In O. Scherzer, editor, *Handbook of Mathematical Methods in Imaging*. Springer, to appear.

[10] S. Foucart and M.J. Lai. Sparsest solutions of underdetermined linear systems via lq-minimization for $0 < q < p$. *Applied and Computational Harmonic Analysis*, 26(3):395–407, 2009.

[11] V. Koltchinskii. The Dantzig selector and sparsity oracle inequalities. *Bernoulli*, 15(3):799–828, 2009.

[12] H. Rauhut. Compressive Sensing and Structured Random Matrices. *Theoretical Foundations and Numerical Methods for Sparse Recovery*, 9, 2010.

[13] H. Rauhut and R. Ward. Sparse legendre expansions via $l_1$ minimization. *Arxiv preprint arXiv:1003.0251*, 2010.

[14] M. Rudelson and R. Vershynin. On sparse reconstruction from Fourier and Gaussian measurements. *Communications on Pure and Applied Mathematics*, 61(8):1025–1045, 2008.

[15] Robert Tibshirani. Regression shrinkage and selection via the Lasso. *Journal of the Royal Statistical Society, Series B*, 58:267–288, 1994.

[16] Sara A. van de Geer. The deterministic lasso. Seminar für Statistik, Eidgenössische Technische Hochschule (ETH) Zürich, 2007.

[17] Sara A. van de Geer and Peter Buhlmann. On the conditions used to prove oracle results for the lasso. *Electronic Journal of Statistics*, 3:1360–1392, 2009.

[18] P. Zhao and B. Yu. On model selection consistency of Lasso. *The Journal of Machine Learning Research*, 7:2563, 2006.

